# Semi-Supervised Learning with Trees

**Charles Kemp, Thomas L. Griffiths, Sean Stromsten & Joshua B. Tenenbaum**
Department of Brain and Cognitive Sciences, MIT, Cambridge, MA 02139
{ckemp,gruffydd,sean_s,jbt}@mit.edu

## Abstract

We describe a nonparametric Bayesian approach to generalizing from few labeled examples, guided by a larger set of unlabeled objects and the assumption of a latent tree-structure to the domain. The tree (or a distribution over trees) may be inferred using the unlabeled data. A prior over concepts generated by a mutation process on the inferred tree(s) allows efficient computation of the optimal Bayesian classification function from the labeled examples. We test our approach on eight real-world datasets.

## 1 Introduction

People have remarkable abilities to learn concepts from very limited data, often just one or a few labeled examples per class. Algorithms for *semi-supervised learning* try to match this ability by extracting strong inductive biases from a much larger sample of *unlabeled* data. A general strategy is to assume some latent structure $\mathcal{T}$ that underlies both the label vector $Y$ to be learned and the observed features $X$ of the full data (unlabeled and labeled; see Figure 1). The unlabeled data can be used to help identify the latent structure $\mathcal{T}$, and an assumption that $Y$ is somehow "smooth" with respect to $\mathcal{T}$ – or in Bayesian terms, can be assigned a strong prior conditional on $\mathcal{T}$ – provides the inductive bias needed to estimate $Y$ successfully from very few labeled examples $Y_{obs}$.

Different existing approaches can be understood within this framework. The closest to our current work is [1] and its cousins [2-5]. The structure $\mathcal{T}$ is assumed to be a low-dimensional manifold, whose topology is approximated by a sparse neighborhood graph defined over the data points (based on Euclidean distance between feature vectors in the $X$ matrix). The label vector $Y$ is assumed to be smooth with respect to $\mathcal{T}$; [1] implements this smoothness assumption by defining a Gaussian field over all complete labelings $Y$ of the neighborhood graph that expects neighbors to have the same label. This approach performs well in classifying data with a natural manifold structure, e.g., handwritten digits.

The graphical model in Figure 1 suggests a more general strategy for exploiting other kinds of latent structure $\mathcal{T}$, not just low-dimensional manifolds. In particular, trees arise prominently in both natural and human-generated domains (e.g., in biology, language and information retrieval). Here we describe an approach to semi-supervised learning based on mapping the data onto the leaf nodes of a rooted (and typically ultrametric) tree $\mathcal{T}$.

The label vector $Y$ is generated from a stochastic mutation process operating over branches of $\mathcal{T}$. Tree $\mathcal{T}$ can be inferred from unlabeled data using either bottom-up methods (agglomerative clustering) or more complex probabilistic methods. The mutation process defines

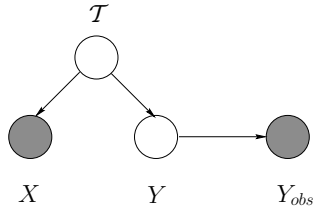

Figure 1: A general approach to semi-supervised learning. $X$ is an observed object-feature matrix, $Y$ the hidden vector of true labels for these objects and $Y_{obs}$ a sparse vector of observed labels. The unlabeled data in $X$ assist in inferring $Y$ by allowing us to infer some latent structure $\mathcal{T}$ that is assumed to generate both $X$ and $Y$.

a prior over all possible labelings of the unlabeled data, favoring those that maximize a tree-specific notion of "smoothness". Figure 2 illustrates this *Tree-Based Bayes (TBB)* approach. Each of the 32 objects in this dataset has two continuous features (x and y coordinates); $X$ is a 32-by-2 matrix. $Y_{obs}$ contains four entries, two positive and two negative. The shading in part (b) represents a probabilistic inference about $Y$: the darker an object's node in the tree, the more likely that its label is positive.

TBB classifies unlabeled data by integrating over all possible labelings of the domain that are consistent with the observed labels $Y_{obs}$, and is thus an instance of optimal Bayesian concept learning [6]. Typically, optimal Bayes is of theoretical interest only [7], because the sum over labelings is in general intractable and it is difficult to specify sufficiently powerful and noise-resistant priors for real-world domains. Here, a prior defined in terms of a tree-based mutation process makes the approach efficient and empirically successful.

The next section describes TBB, as well as a simple heuristic method, *Tree Nearest Neighbor (TNN)*, which we show approximates TBB in the limit of high mutation rate. Section 3 presents experimental comparisons with other approaches on a range of datasets.

(a)         (b)

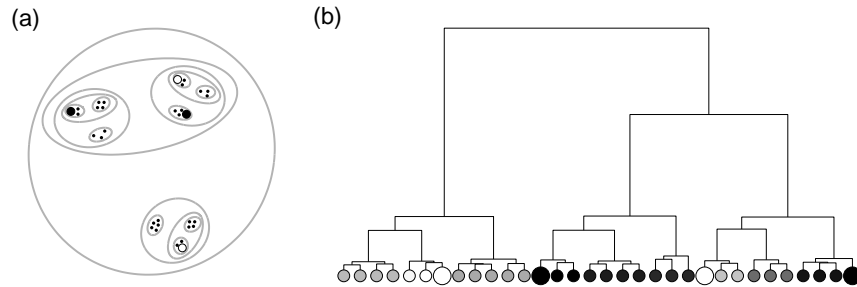

Figure 2: Illustration of the Tree-Based Bayesian approach to semi-supervised learning. (a) We observe a set of unlabeled objects (small points) with some latent hierarchical structure (gray ellipses) along with two positive and two negative examples of a new concept (black and white circles). (b) Inferring the latent tree, and treating the concept as generated from a mutation process on the tree, we can probabilistically classify the unlabeled objects.

## 2 Tree-Based Bayes (TBB)

We assume a binary classification problem with $Y \in \{-1, 1\}^n$. We choose a label $y_i$ for unlabeled object $x_i$ by computing $p(y_i = 1|Y_{obs}, X)$ and thresholding at 0.5. Generalization to the multi-class case will be straightforward.

Ideally we would sum over all possible latent trees $\mathcal{T}$:

$$p(y_i = 1|Y_{obs}, X) = \sum_{\mathcal{T}} p(y_i = 1|Y_{obs}, \mathcal{T})p(\mathcal{T}|Y_{obs}, X) \qquad (1)$$

First we consider $p(y_i = 1|Y_{obs}, \mathcal{T})$ and the classification of object $x_i$ given a particular tree $\mathcal{T}$. Section 2.2 discusses $p(\mathcal{T}|Y_{obs}, X)$, the inference of tree $\mathcal{T}$, and approaches to approximating the sum over trees in Equation 1.

We predict object $x_i$'s label by summing over all possible complete labelings $Y$ of the data:

$$p(y_i = 1|Y_{obs}, \mathcal{T}) = \sum_Y p(y_i = 1|Y)p(Y|Y_{obs}, \mathcal{T}) \tag{2}$$

$$= \sum_Y \frac{p(y_i = 1|Y)p(Y_{obs}|Y, \mathcal{T})p(Y|\mathcal{T})}{p(Y_{obs}|\mathcal{T})} \tag{3}$$

$$= \frac{\sum_Y p(y_i = 1|Y)p(Y_{obs}|Y)p(Y|\mathcal{T})}{\sum_Y p(Y_{obs}|Y)p(Y|\mathcal{T})} \tag{4}$$

In general, the likelihood $p(Y_{obs}|Y)$ depends on assumptions about sampling and noise. Typical simplifying assumptions are that the labeled objects were chosen randomly from all objects in the domain, and that all observations are free of noise. Then $p(Y_{obs}|Y) \propto 1$ if $Y_{obs}$ is consistent with $Y$ and is zero otherwise.

Under these assumptions, Equation 4 becomes:

$$p(y_i = 1|Y_{obs}, \mathcal{T}) = \frac{\sum_{Y \text{ consistent with } Y_{obs}:y_i=1} p(Y|\mathcal{T})}{\sum_{Y \text{ consistent with } Y_{obs}} p(Y|\mathcal{T})} \tag{5}$$

The probability that $y_i = 1$ reduces to the weighted fraction of label vectors consistent with $Y_{obs}$ that set $y_i = 1$, with each label vector weighted by its prior under the tree, $p(Y|\mathcal{T})$.

When class frequencies are unbalanced, small training sets provide little scope for learning if constructed using random sampling. Consider the problem of identifying genetic markers for a disease that afflicts one person in 10,000. A training set for this problem might be constructed by "retrospective sampling," e.g. taking data from 20 patients with the disease and 20 healthy subjects. Randomly sampling subjects from the entire population would mean that even a medium-sized training set would have little chance of including anyone with the disease.

Retrospective sampling can be modeled by specifying a more complex likelihood $p(Y_{obs}|Y)$. The likelihood can also be modified to handle additional complexities, such as learning from labeled examples of just a single class, or learning in the presence of label noise. We consider none of these complexities here. Our experiments explore both random and retrospective sampling, but the algorithm we implement is strictly correct only for noise-free learning under random sampling.

## 2.1 Bayesian classification with a mutation model

In many tree-structured domains it is natural to think of features arising from a history of stochastic events or mutations. We develop a mutation model that induces a sensible "smoothness" prior $p(Y|\mathcal{T})$ and enables efficient computation of Equation 5 via belief propagation on a Bayes net. The model combines aspects of several previous proposals for probabilistic learning with trees [8, 9, 10].

Let $L$ be a feature corresponding to the class label. Suppose that $L$ is defined at every point along every branch, not just at the leaf nodes where the data points lie. Imagine $L$ spreading out over the tree from root to leaves — it starts out at the root with some value and could switch values at any point along any branch. Whenever a branch splits, both lower branches inherit the value of $L$ at the point immediately before the split.

Transitions between states of $L$ are modeled using a continuous-time Markov chain with infinitesimal matrix:

$$Q = \left[ \begin{array}{cc} -\lambda & \lambda \\ \lambda & -\lambda \end{array} \right]$$

The free parameter, $\lambda$, will be called the mutation rate. Note that the mutation process is symmetric: mutations from -1 to 1 are just as likely as mutations in the other direction. Other models of mutation could be substituted if desired. Generalization to the $k$-class case is achieved by specifying a $k$ by $k$ matrix $Q$, with $-\lambda$ on the diagonal and $\frac{\lambda}{k-1}$ on the off-diagonal.

Transition probabilities along a branch of length $t$ are given by:

$$e^{Qt} = \left[ \begin{array}{cc} \frac{1+e^{-2\lambda t}}{2} & \frac{1-e^{-2\lambda t}}{2} \\ \frac{1-e^{-2\lambda t}}{2} & \frac{1+e^{-2\lambda t}}{2} \end{array} \right] \tag{6}$$

That is, the probability that a parent and child separated by a branch of length $t$ have different values of $L$ is $\frac{1-e^{-2\lambda t}}{2}$.

This mutation process induces a prior $p(Y|\mathcal{T})$ equal to the probability of generating the label vector $Y$ over leaves of $\mathcal{T}$ under the mutation process. The resulting distribution favors labelings that are "smooth" with respect to $\mathcal{T}$. Regardless of $\lambda$, it is always more likely for $L$ to stay the same than to switch its value along a branch. Thus labelings that do not require very many mutations are preferred, and the two hypotheses that assign the same label to all leaf nodes receive the most weight. Because mutations are more likely to occur along longer branches, the prior also favors hypotheses in which label changes occur between clusters (where branches tend to be longer) rather than within clusters (where branches tend to be shorter).

The independence assumptions implicit in the mutation model allow the right side of Equation 5 to be computed efficiently. Inspired by [9], we set up a Bayes net with the same topology as $\mathcal{T}$ that captures the joint probability distribution over all nodes. We associate with each branch a conditional probability table that specifies the value of the child conditioned on the value of the parent (based on Equation 6), and set the prior probabilities at the root node to the uniform distribution (the stationary distribution of the Markov chain specified by $Q$). Evaluating Equation 5 now reduces to a standard problem of inference in a Bayes net – we clamp the nodes in $Y_{obs}$ to their observed values, and compute the posterior marginal probability at node $y_i$. The tree structure makes this computation efficient and allows specially tuned inference algorithms, as in [9].

## 2.2 A distribution over trees

We now consider $p(\mathcal{T}|Y_{obs}, X)$, the second component of Equation 1. Using Bayes' theorem:

$$p(\mathcal{T}|Y_{obs}, X) \propto p(Y_{obs}, X|\mathcal{T})p(\mathcal{T}) \tag{7}$$

We assume that each discrete feature in $X$ is generated independently over $\mathcal{T}$ according to the mutation model just outlined. Continuous features can be handled by an analogous stochastic diffusion process in a continuous space (see for example [11]). Because the features are conditionally independent of each other and of $Y_{obs}$ given the tree, $p(Y_{obs}, X|\mathcal{T})$ can be computed using the methods of the previous section.

To finish the theoretical development of the model it remains only to specify $p(\mathcal{T})$, a prior over tree structures. Section 3.2 uses a uniform prior, but a Dirichlet Diffusion Tree prior is another option [11].

### 2.3 Approximating the sum over trees

The sum over trees in Equation 1 is intractable for datasets of even moderate size. We therefore consider two approximations. Markov Chain Monte Carlo (MCMC) techniques have been used to approximate similar sums over trees in Bayesian phylogenetics [12], and Section 3.2 applies these ideas to a small-scale example. Although theoretically attractive, MCMC approaches are still expensive to use with large datasets. Section 3.1 follows a simpler approach: we assume that most of the probability $p(\mathcal{T}|Y_{obs}, X)$ is concentrated on or near the most probable tree $\mathcal{T}^*$ and approximate Equation 1 as $p(y_i = 1|Y_{obs}, \mathcal{T}^*)$. The tree $\mathcal{T}^*$ can be estimated using more or less sophisticated means. In Section 3.1 we use a greedy method – average-link agglomerative clustering on the object-feature matrix $X$, using Hamming or Euclidean distance in discrete or continuous domains, respectively. In Section 3.2 we compare this greedy method to the best tree found in our MCMC runs. Note that we ignore $Y_{obs}$ when building $\mathcal{T}^*$, because we run many trials on each dataset and do not want to compute a new tree for each value of $Y_{obs}$. Since our data include many features and few labeled objects, the contribution of $Y_{obs}$ is likely to be negligible.

### 2.4 Tree Nearest Neighbor (TNN)

A Bayesian formulation based on the mutation process provides a principled approach to learning with trees, but there are simpler algorithms that instantiate similar intuitions. For instance, we could build a one-nearest-neighbor classifier using the metric of distance in the tree $\mathcal{T}$ (with ties resolved randomly). It is clear how this *Tree Nearest Neighbor (TNN)* algorithm reflects the assumption that nearby leaves in $\mathcal{T}$ are likely to have the same label, but it is not necessarily clear when and why this simple approach should work well.

An analysis of Tree-Based Bayes provides some insight here – TBB and TNN become equivalent when the $\lambda$ parameter of TBB is set sufficiently high.

**Theorem 1** *For each ultrametric tree $\mathcal{T}$, there is a $\lambda_0$ such that TNN and TBB produce identical classifications for all examples with a unique nearest neighbor when $\lambda > \lambda_0$.*

A proof is available at `http://www.mit.edu/~ckemp/papers/treesslproof.pdf`, but we give some intuition for the result here. Consider the Bayes net described in Section 2.1 and suppose $x_i$ is an unlabeled object. The value chosen for $y_i$ will depend on all the labels in $Y_{obs}$, but the influence of any single label decreases with distance in the tree from $y_i$. Once $\lambda$ becomes sufficiently high it can be shown that $y_i$ is always determined uniquely by the closest labeled example in the tree.

Given this equivalence between the algorithms, TNN is the method of choice when a high mutation rate is indicated. It is not only faster, but numerically more stable. For large values of $\lambda$, the probabilities manipulated by TBB become very close to 0.5 and variables that should be different may become indistinguishable within the limits of computational precision. Our implementation of TBB therefore uses TNN when cross-validation indicates that a sufficiently high value of $\lambda$ is required.

## 3 Experiments

### 3.1 Trees versus Manifolds

We compared TBB and TNN with the Laplacian method of Belkin and Niyogi [4], an approach that effectively assumes a latent manifold structure $\mathcal{T}$. We also ran generic one-nearest neighbor (NN) as a baseline.

The best performing method on a given dataset should be the algorithm that assumes the

right latent structure for that domain. We therefore tested the algorithms on several different types of data: four taxonomic datasets (Beetles, Crustaceans, Salamanders and Worms, with 192, 56, 30 and 286 objects respectively), two molecular biology sets (Gene Promoter and Gene Splice, with sizes 106 and 3190), and two "manifold" sets (Digits and Vowels, with sizes 10,000 and 990).

The taxonomic datasets were expected to have a tree-like structure. Each set describes the external anatomy of a group of species, based on data available at `http://biodiversity.uno.edu/delta/`. One feature in the Beetles set, for example, indicates whether a beetle's body is "strongly flattened, slightly flattened to moderately convex, or strongly convex." Since these taxonomic sets do not include class labels, we chose features at random to stand in for the class label. We averaged across five such choices for each dataset.

The molecular biology sets were taken from the UCI repository. The objects in both sets are strings of DNA, and tree structures might also be appropriate here since these strings arose through evolution. The manifold sets arose from human motor behaviors, and were therefore expected to have a low-dimensional manifold structure. The Digits data are a subset of the MNIST data, and the Vowels data are taken from the UCI repository.

Our experiments focused on learning from very small labeled sets. The number of labeled examples was always set to a small multiple ($m = 1, 2, 3, 5,$ or $10$) of the total number of classes. The algorithms were compared under random and retrospective sampling, and training sets were always sampled with replacement. For each training-set size $m$, we averaged across 10 values of $Y_{obs}$ obtained by randomly sampling from the vector $Y$. Free parameters for TBB ($\lambda$) and Laplacian (number of nearest neighbors, number of eigenvectors) were chosen using randomized leave-one-out cross-validation.

Figure 3a shows the performance of the algorithms under random sampling for four representative datasets. TBB outperforms the other algorithms across the four taxonomic sets (only Beetles and Crustaceans shown), but the differences between TBB and Nearest Neighbor are rather small. These results do suggest a substantial advantage for TBB over Laplacian in tree-structured domains. As expected, this pattern is reversed on the Digits set, but it is encouraging that the tree-based methods can still improve on Nearest Neighbor even for datasets that are not normally associated with trees. Neither method beats the baseline on the Vowels or the Gene Promoter sets, but TBB performs well on the Gene Splice set, which suggests that it may find further uses in computational biology.

More dramatic differences between the algorithms appear under retrospective sampling (Figure 3b). There is a clear advantage here for TBB on the taxonomic sets. TBB fares better than the other algorithms when the class proportions in the training set do not match the proportions in the population, and it turns out that many of the features in the taxonomic datasets are unbalanced. Since the other datasets have classes of approximately equal size, the results for retrospective sampling are similar to those for random sampling.

While not conclusive, our results suggest that TBB may be the method of choice on tree-structured datasets, and is robust even for datasets (like Digits) that are not clearly tree-structured.

## 3.2   MCMC over trees

Figure 3 shows that TBB can perform well on real-world datasets using only a single tree. Working with a distribution over trees, although costly, could improve performance when there is not sufficient data to strongly constrain the best tree, or when the domain is not strongly tree-structured. Using a small synthetic example, we explored one such case: learning from very sparse and noisy data in a tree-structured domain.

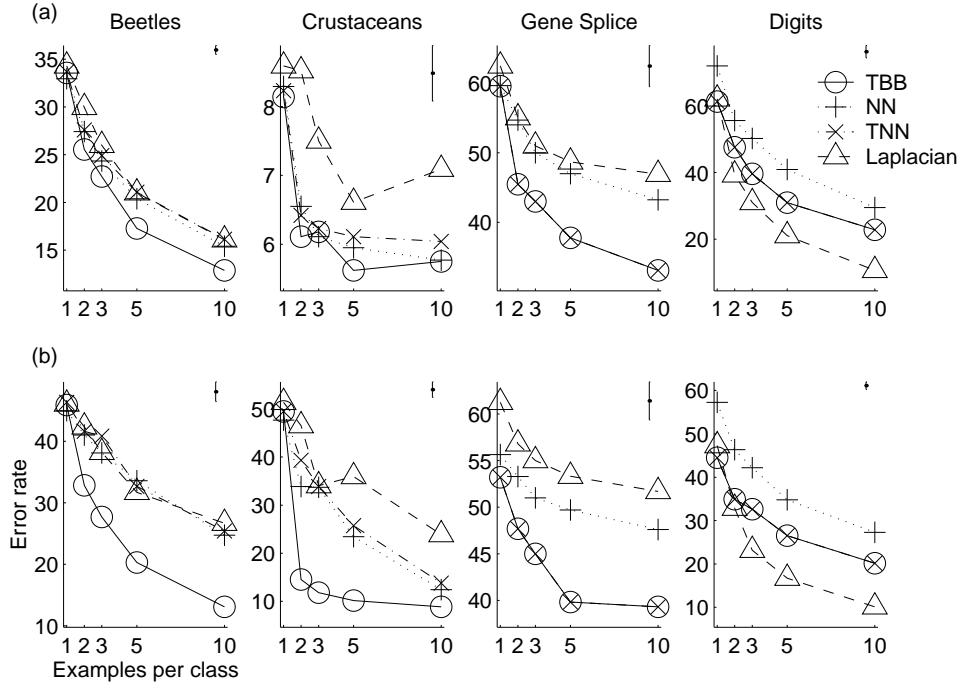

Figure 3: Error rates for four datasets under (a) random and (b) retrospective sampling, as a function of the number of labeled examples $m$ per class. Mean standard error bars for each dataset are shown in the upper right corner of the plot.

We generated artificial datasets consisting of 20 objects. Each dataset was based on a "true" tree $\mathcal{T}_0$, with objects at the leaves of $\mathcal{T}_0$. Each object was represented by a vector of 20 binary features generated by a mutation process over $\mathcal{T}_0$, with high $\lambda$. Most feature values were missing; the algorithms saw only 5 of the 20 features for each object. For each dataset, we created 20 test concepts from the same mutation process. The algorithms saw $m$ labeled examples of each test concept and had to infer the labels of the remaining objects. This experiment was repeated for 10 random trees $\mathcal{T}_0$.

Our MCMC approach was inspired by an algorithm for reconstruction of phylogenetic trees [12], which uses Metropolis-Hastings over tree topologies with two kinds of proposals: local (nearest neighbor interchange) and global (subtree pruning and regrafting). Unlike the previous section, none of the trees considered (including the true tree $\mathcal{T}_0$) was ultrametric. Instead, each branch in each tree was assigned a fixed length. This meant that any two trees

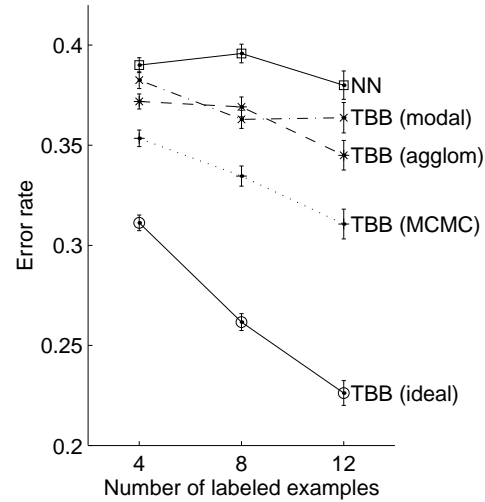

Figure 4: Error rates on sparse artificial data as a function of number of labels observed.

with the same hierarchical structure were identical, and we did not have to store trees with the same topology but different branch lengths.

Figure 4 shows the mean classification error rate, based on 1600 samples after a burn-in of 400 iterations. Four versions of TBB are shown: "ideal" uses the true tree $\mathcal{T}_0$, "MCMC" uses model averaging over a distribution of trees, "modal" uses the single most likely tree in the distribution, and "agglom" uses a tree built by average-link clustering. The ideal learner beats all others because the true tree is impossible to identify with such sparse data. Using MCMC over trees brings TBB substantially closer to the ideal than simpler alternatives that ignore the tree structure (NN) or consider only a single tree (modal, agglom).

## 4    Conclusion

We have shown how to make optimal Bayesian concept learning tractable in a semi-supervised setting by assuming a latent tree structure that can be inferred from the unlabeled data and defining a prior for concepts based on a mutation process over the tree. Our Bayesian framework supports many possible extensions, including active learning, feature selection, and model selection. Inferring the nature of the latent structure $\mathcal{T}$ – rather than assuming a manifold structure or a tree structure – is a particularly interesting problem. When little is known about the form of $\mathcal{T}$, Bayesian methods for model selection could be used to choose among approaches that assume manifolds, trees, flat clusters, or other canonical representational forms.

**Acknowledgments** This project was supported by the DARPA CALO program and NTT Communication Science Laboratories. Our implementation of the Laplacian method was based on code provided by Mikhail Belkin.

## References

[1]  X. Zhu, Z. Ghahramani, and J. Lafferty. Semi-supervised learning using Gaussian fields and harmonic functions. In *ICML*, volume 20, 2003.

[2]  M. Szummer and T. Jaakkola. Partially labeled classification with Markov random walks. In *NIPS*, volume 14, 2002.

[3]  A. Blum and S. Chawla. Learning from labeled and unlabeled data using graph mincuts. In *ICML*, volume 18, 2001.

[4]  M. Belkin and P. Niyogi. Semi-supervised learning on manifolds. 2003. To appear in *Machine Learning*, Special Issue on Theoretical Advances in Data Clustering.

[5]  O. Chapelle, J. Weston, and B. Schölkopf. Cluster kernels for semi-supervised learning. In *NIPS*, volume 15, 2003.

[6]  T. M. Mitchell. *Machine Learning*. McGraw-Hill, 1997.

[7]  D. Haussler, M. Kearns, and R. Schapire. Bounds on the sample complexity of Bayesian learning using information theory and the VC dimension. *Machine Learning*, 14(1), 1994.

[8]  C. Kemp and J. B. Tenenbaum. Theory-based induction. In *Proceedings of the 25th Annual Conference of the Cognitive Science Society*, 2003.

[9]  L. Shih and D. Karger. Learning classes correlated to a hierarchy. 2003. Unpublished manuscript.

[10]  J.-P. Vert. A tree kernel to analyze phylogenetic profiles. *Bioinformatics*, 1(1):1–9, 2002.

[11]  R. Neal. Defining priors for distributions using Dirichlet diffusion trees. Technical Report 0108, University of Toronto, 2001.

[12]  H. Jow, C. Hudelot, M. Rattray, and P. Higgs. Bayesian phylogenetics using an RNA substitution model applied to early mammalian evolution. *Molecular Biology and Evolution*, 19(9):1951–1601, 2002.